# STABILITY RESULTS FOR NEURAL NETWORKS

A. N. Michel[1] , J. A. Farrell[1] , and W. Porod[2]

Department of Electrical and Computer Engineering
University of Notre Dame
Notre Dame, IN 46556

## ABSTRACT

In the present paper we survey and utilize results from the qualitative theory of large scale interconnected dynamical systems in order to develop a qualitative theory for the Hopfield model of neural networks. In our approach we view such networks as an interconnection of many single neurons. Our results are phrased in terms of the qualitative properties of the individual neurons and in terms of the properties of the interconnecting structure of the neural networks. Aspects of neural networks which we address include asymptotic stability, exponential stability, and instability of an equilibrium; estimates of trajectory bounds; estimates of the domain of attraction of an asymptotically stable equilibrium; and stability of neural networks under structural perturbations.

## INTRODUCTION

In recent years, neural networks have attracted considerable attention as candidates for novel computational systems[1-3]. These types of large-scale dynamical systems, in analogy to biological structures, take advantage of distributed information processing and their inherent potential for parallel computation[4,5]. Clearly, the design of such neural-network-based computational systems entails a detailed understanding of the dynamics of large-scale dynamical systems. In particular, the stability and instability properties of the various equilibrium points in such networks are of interest, as well as the extent of associated domains of attraction (basins of attraction) and trajectory bounds.

In the present paper, we apply and survey results from the qualitative theory of large scale interconnected dynamical systems[6-9] in order to develop a qualitative theory for neural networks. We will concentrate here on the popular Hopfield model[3], however, this type of analysis may also be applied to other models. In particular, we will address the following problems: (i) determine the stability properties of a given equilibrium point; (ii) given that a specific equilibrium point of a neural network is asymptotically stable, establish an estimate for its domain of attraction; (iii) given a set of initial conditions and external inputs, establish estimates for corresponding trajectory bounds; (iv) give conditions for the instability of a given equilibrium point; (v) investigate stability properties under structural perturbations. The present paper contains local results. A more detailed treatment of local stability results can be found in Ref. 10, whereas global results are contained in Ref. 11.

In arriving at the results of the present paper, we make use of the method of analysis advanced in Ref. 6. Specifically, we view high dimensional neural network as an

interconnection of individual subsystems (neurons). This *interconnected systems view-point* makes our results distinct from others derived in the literature[1,12]. Our results are phrased in terms of the qualitative properties of the free subsystems (individual neurons, disconnected from the network) and in terms of the properties of the interconnecting structure of the neural network. As such, these results may constitute useful design tools. This approach makes possible the systematic analysis of high dimensional complex systems and it frequently enables one to circumvent difficulties encountered in the analysis of such systems by conventional methods.

The structure of this paper is as follows. We start out by defining the Hopfield model and we then introduce the interconnected systems viewpoint. We then present representative stability results, including estimates of trajectory bounds and of domains of attraction, results for instability, and conditions for stability under structural perturbations. Finally, we present concluding remarks.

## THE HOPFIELD MODEL FOR NEURAL NETWORKS

In the present paper we consider neural networks of the Hopfield type[3]. Such systems can be represented by equations of the form

$$\dot{u}_i = -b_i u_i + \sum_{j=1}^{N} A_{ij} \, G_j(u_j) + U_i(t), \quad for \;\; i = 1, \ldots, N, \qquad (1)$$

where $A_{ij} = \frac{T_{ij}}{C_i}, U_i(t) = \frac{I_i(t)}{C_i}$ and $b_i = \frac{1}{\tau_i C_i}$. As usual, $C_i > 0, T_{ij} = \frac{1}{R_{ij}}, R_{ij} \epsilon R = (-\infty, \infty), \frac{1}{\tau_i} = \frac{1}{R_i} + \sum_{j=1}^{N} |T_{ij}|$, $R_i > 0, I_i : R^+ = [0, \infty) \to R, I_i$ is continuous, $\dot{u}_i = \frac{du_i}{dt}, G_i : R \to (-1, 1), G_i$ is continuously differentiable and strictly monotonically increasing (i.e., $G_i(u_i') > G_i(u_i'')$ if and only if $u_i' > u_i''$), $u_i G_i(u_i) > 0$ for all $u_i \neq 0$, and $G_i(0) = 0$. In (1), $C_i$ denotes capacitance, $R_{ij}$ denotes resistance (possibly including a sign inversion due to an inverter), $G_i(\cdot)$ denotes an amplifier nonlinearity, and $I_i(\cdot)$ denotes an external input.

In the literature it is frequently assumed that $T_{ij} = T_{ji}$ for all $i, j = 1, \ldots, N$ and that $T_{ii} = 0$ for all $i = 1, \ldots, N$. We will make these assumptions only when explicitly stated.

We are interested in the qualitative behavior of solutions of (1) near equilibrium points (rest positions where $\dot{u}_i \equiv 0, \; for \; i = 1, \ldots, N$). By setting the external inputs $U_i(t), \; i = 1, \ldots, N$, equal to zero, we define $u^* = [u_1^*, \ldots, u_N^*]^T \epsilon R^N$ to be an *equilibrium* for (1) provided that $-b_i u_i^* + \sum_{j=1}^{N} A_{ij} \, G_j(u_j^*) = 0, \; for \; i = 1, \ldots, N$. The locations of such equilibria in $R^N$ are determined by the interconnection pattern of the neural network (i.e., by the parameters $A_{ij}, i, j = 1, \ldots, N$) as well as by the parameters $b_i$ and the nature of the nonlinearities $G_i(\cdot), i = 1, \ldots, N$.

Throughout, we will assume that a given equilibrium $u^*$ being analyzed is an *isolated* equilibrium for (1), i.e., there exists an $r > 0$ such that in the neighborhood $B(u^*, r) = \{(u - u^*) \epsilon R^N : |u - u^*| < r\}$ no equilibrium for (1), other than $u = u^*$, exists.

When analyzing the stability properties of a given equilibrium point, we will be able to assume, without loss of generality, that this equilibrium is located at the origin $u = 0$ of $R^N$. If this is not the case, a trivial transformation can be employed which shifts the equilibrium point to the origin and which leaves the structure of (1) the same.

## INTERCONNECTED SYSTEMS VIEWPOINT

We will find it convenient to view system (1) as an interconnection of $N$ *free subsystems* (or *isolated subsystems*) described by equations of the form

$$\dot{p}_i = -b_i p_i + A_{ii}\, G_i(p_i) + U_i(t).$$  (2)

Under this viewpoint, the *interconnecting structure* of the system (1) is given by

$$G_i(x_1,\ldots,x_n) \triangleq \sum_{\substack{j=1 \\ i \neq j}}^{N} A_{ij} G_j(x_j), \quad i = 1,\ldots,N.$$  (3)

Following the method of analysis advanced in[6], we will establish stability results which are phrased in terms of the qualitative properties of the free subsystems (2) and in terms of the properties of the interconnecting structure given in (3). This method of analysis makes it often possible to circumvent difficulties that arise in the analysis of complex high-dimensional systems. Furthermore, results obtained in this manner frequently yield insight into the dynamic behavior of systems in terms of system components and interconnections.

## GENERAL STABILITY CONDITIONS

We demonstrate below an example of a result for exponential stability of an equilibrium point. The principal Lyapunov stability results for such systems are presented, e.g., in Chapter 5 of Ref. 7.

We will utilize the following hypotheses in our first result.

**(A-1)** For system (1), the external inputs are all zero, i.e.,

$$U_i(t) \equiv 0, \quad i = 1,\ldots,N.$$

**(A-2)** For system (1), the interconnections satisfy the estimate

$$x_i A_{ij}\, G_j(x_j) \leq x_i\, a_{ij} x_j$$

for all $|x_i| < r_i$, $|x_j| < r_j$, $i,j = 1,\ldots,N$, where the $a_{ij}$ are real constants.

**(A-3)** There exists an $N$-vector $\alpha > 0$ (i.e., $\alpha^T = (\alpha_1,\ldots,\alpha_N)$ and $\alpha_i > 0$, *for all* $i = 1,\ldots,N$) such that the *test matrix* $S = [s_{ij}]$

$$s_{ij} = \begin{cases} \alpha_i(-b_i + a_{ii}), & i = j \\ (\alpha_i\, a_{ij} + \alpha_j\, a_{ji})/2, & i \neq j \end{cases}$$

*is negative definite, where the $b_i$ are defined in (1) and the $a_{ij}$ are given in (A-2).*

We are now in a position to state and prove the following result.

**Theorem 1** *The equilibrium $x = 0$ of the neural network (1) is **exponentially stable** if hypotheses (A-1), (A-2) and (A-3) are satisfied.*

**Proof.** For (1) we choose the Lyanpunov function

$$v(x) = \sum_{i=1}^{N} \frac{1}{2} \alpha_i x_i^2 \tag{4}$$

where the $\alpha_i$ are given in (A-3). This function is clearly positive definite. The time derivative of $v$ along the solutions of (1) is given by

$$Dv_{(1)}(x) = \sum_{i=1}^{N} \frac{1}{2} \alpha_i (2x_i)[-b_i x_i + \sum_{j=1}^{N} A_{ij} G_j(x_j)]$$

where (A-1) has been invoked. In view of (A-2) we have

$$Dv_{(1)}(x) \leq \sum_{i=1}^{N} \alpha_i(-b_i x_i^2 + x_i \sum_{j=1}^{N} a_{ij} x_j)$$

$$= x^T R x \quad \text{for all } |x|_2 < r$$

where $r = \min_i(r_i)$, $|x|_2 = \left( \sum_{i=1}^{N} x_i^2 \right)^{1/2}$, and the matrix $R = [r_{ij}]$ is given by

$$r_{ij} = \begin{cases} \alpha_i(-b_i + a_{ii}), & i = j \\ \alpha_i a_{ij}, & i \neq j. \end{cases}$$

But it follows that

$$x^T R x = x^T \left( \frac{R + R^T}{2} \right) x = x^T S x \leq \lambda_M(S) |x|_2^2 \tag{5}$$

where $S$ is the matrix given in (A-3) and $\lambda_M(S)$ denotes the largest eigenvalue of the real symmetric matrix $S$. Since $S$ is by assumption negative definite, we have $\lambda_M(S) < 0$. It follows from (4) and (5) that in some neighborhood of the origin $x = 0$, we have $c_1|x|_2^2 \leq v(x) \leq c_2|x|_2^2$ and $Dv_{(1)}(x) \leq -c_3|x|_2^2$, where $c_1 = \frac{1}{2}\min_i \alpha_i > 0$, $c_2 = \frac{1}{2}\max_i \alpha_i > 0$, and $c_3 = -\lambda_M(S) > 0$. Hence, the equilibrium $x = 0$ of the neural network (1) is exponentially stable (c.f. Theorem 9.10 in Ref. 7).

Consistent with the philosophy of viewing the neural network (1) as an interconnection of $N$ free subsystems (2), we think of the Lyapunov function (4) as consisting of a weighted sum of Lyapunov functions for each free subsystem (2) (with $U_i(t) \equiv 0$). The weighting vector $\alpha > 0$ provides flexibility to emphasize the relative importance of the qualitative properties of the various individual subsystems. Hypothesis $(A-2)$ provides a measure of interaction between the various subsystems (3). Furthermore, it is emphasized that Theorem 1 does not require that the parameters $A_{ij}$ in (1) form a symmetric matrix.

## WEAK COUPLING CONDITIONS

The test matrix S given in hypothesis $(A-3)$ has off-diagonal terms which may be positive or nonpositive. For the special case where the off-diagonal terms of the test matrix $S = [s_{ij}]$ are non-negative, equivalent stability results may be obtained which are much easier to apply than Theorem 1. Such results are called *weak-coupling conditions* in the literature[6,9]. The conditions $s_{ij} \geq 0$ for all $i \neq j$ may reflect properties of the system (1) or they may be the consequence of a majorization process.

In the proof of the subsequent result, we will make use of some of the properties of M- matrices (see, for example, Chapter 2 in Ref. 6). In addition we will use the following assumptions.

**(A-4)** For system (1), the nonlinearity $G_i(x_i)$ satisfies the sector condition

$$0 < \sigma_{i1} \leq \frac{G_i(x_i)}{x_i} \leq \sigma_{i2}, \quad for\ all\ |x_i| < r_i, \quad i = 1, \ldots, N.$$

**(A-5)** The successive principal minors of the $N \times N$ *test matrix* $D = [d_{ij}]$

$$d_{ij} = \begin{cases} \frac{b_i}{\sigma_{i2}} - A_{ii}, & i = j \\ -|A_{ij}|, & i \neq j \end{cases}$$

*are all positive where, the $b_i$ and $A_{ij}$ are defined in (1) and $\sigma_{i2}$ is defined in $(A-4)$.*

**Theorem 2** *The equilibrium $x = 0$ of the neural network (1) is* **asymptotically stable** *if hypotheses (A-1), (A-4) and (A-5) are true.*

**Proof.** The proof proceeds[10] along lines similar to the one for Theorem 1, this time with the following Lyapunov function

$$v(x) = \sum_{i=1}^{N} \alpha_i |x_i|. \tag{6}$$

The above Lyapunov function again reflects the interconnected nature of the whole system. Note that this Lyapunov function may be viewed as a generalized Hamming distance of the state vector from the origin.

## ESTIMATES OF TRAJECTORY BOUNDS

In general, one is not only interested in questions concerning the stability of an equilibrium of the system (1), but also in performance. One way of assessing the qualitative properties of the neural system (1) is by investigating solution bounds near an equilibrium of interest. We present here such a result by assuming that the hypotheses of Theorem 2 are satisfied.

In the following, we will not require that the external inputs $U_i(t)$, $i = 1, \ldots, N$ be zero. However, we will need to make the additional assumptions enumerated below.

**(A-6)** Assume that there exist $\lambda_i > 0$, *for* $i = 1, \ldots, N$, and an $\epsilon > 0$ such that

$$\left(\frac{b_i}{\sigma_{i2}} - A_{ii}\right) - \sum_{\substack{j=1 \\ i \neq j}}^{N} \left(\frac{\lambda_j}{\lambda_i}\right) |A_{ji}| \geq \epsilon > 0, \quad i = 1, \ldots, N$$

where $b_i$ and $A_{ij}$ are defined in (1) and $\sigma_{i2}$ is defined in (A-4).

**(A-7)** Assume that for system (1),

$$\sum_{i=1}^{N} \lambda_i |U_i(t)| \leq k \quad \text{for all} \quad t \geq 0$$

for some constant $k > 0$ where the $\lambda_i$, $i = 1, \ldots, N$ are defined in (A-6).

In the proof of our next theorem, we will make use of a comparison result. We consider a scalar comparison equation of the form $\dot{y} = G(y)$ where $y \epsilon R, G : B(r) \to R$ for some $r > 0$, and $G$ is continuous on $B(r) = \{x \epsilon R : |x| < r\}$. We can then prove the following auxiliary theorem: Let $p(t)$ denote the maximal solution of the comparison equation with $p(t_0) = y_0 \epsilon B(r)$, $t \geq t_0 > 0$. If $r(t)$, $t \geq t_0 \geq 0$ is a continuous function such that $r(t_0) \leq y_0$, and if $r(t)$ satisfies the differential inequality $Dr(t) = \lim_{k \to 0+} \frac{1}{k} \sup[r(t+k) - r(t)] \leq G(r(t))$ almost everywhere, then $r(t) \leq p(t)$ for $t \geq t_0 \geq 0$, for as long as both $r(t)$ and $p(t)$ exist. For the proof of this result, as well as other comparison theorems, see e.g., Refs. 6 and 7.

For the next theorem, we adopt the following notation. We let $\delta = \min_i \sigma_{i1}$ where $\sigma_{i1}$ is defined in $(A-4)$, we let $c = \epsilon \delta$, where $\epsilon$ is given in (A-6), and we let $\phi(t, t_0, x_0) = [\phi_1(t, t_0, x_0), \ldots, \phi_N(t, t_0, x_0)]^T$ denote the solution of (1) with $\phi(t_0, t_0, x_0) = x_0 = (x_{10}, \ldots, x_{N0})^T$ for some $t_0 \geq 0$.

We are now in a position to prove the following result, which provides bounds for the solution of (1).

**Theorem 3** *Assume that hypotheses (A-6) and (A-7) are satisfied. Then*

$$\|\phi(t, t_0, x_0)\| \triangleq \sum_{i=1}^{N} \lambda_i |\phi_i(t, t_0, x_0)| \leq (\alpha - \frac{k}{c})e^{-c(t-t_0)} + \frac{k}{c}, \quad t \geq t_0 \geq 0$$

*provided that* $\alpha > k/c$ *and* $\|x_0\| = \sum_{i=1}^{N} \lambda_i |x_{i0}| \leq \alpha$, *where the* $\lambda_i$, $i = 1, \ldots, N$ *are given in (A-6) and $k$ is given in (A-7).*

**Proof.** For (1) we choose the Lyapunov function

$$v(x) = \sum_{i=1}^{N} \lambda_i |x_i|. \tag{7}$$

Along the solutions of (1), we obtain

$$Dv_{(1)}(x) \leq \lambda^T Dw + \sum_{i=1}^{N} \lambda_i |U_i(t)| \tag{8}$$

where $w^T = \left[ \frac{G_1(x_1)}{x_1} |x_1|, \ldots, \frac{G_N(x_N)}{x_N} |x_N| \right]$, $\lambda = (\lambda_1, \ldots, \lambda_N)^T$, and $D = [d_{ij}]$ is the test matrix given in (A-5). Note that when (A-6) is satisfied, as in the present theorem, then (A-5) is automatically satisfied. Note also that $w \geq 0$ (i.e., $w_i \geq 0$, $i = 1, \ldots, N$) and $w = 0$ if and only if $x = 0$.

Using manipulations involving (A-6), (A-7) and (8), it is easy to show that $Dv_{(1)}(x) \leq -cv(x) + k$. This inequality yields now the comparison equation $\dot{y} = -cy + k$, whose unique solution is given by

$$p(t, t_0, p_0) = \left( p_0 - \frac{k}{c} \right) e^{-c(t - t_0)} + \frac{k}{c}, \quad \text{for all } t \geq t_0.$$

If we let $r = v$, then we obtain from the comparison result

$$p(t) \geq r(t) = v(\phi(t, t_0, x_0)) = \sum_{i=1}^{N} \lambda_i |\phi_i(t, t_0, x_0)| = \|\phi(t, t_0, x_0)\|,$$

i.e., the desired estimate is true, provided that $|r(t_0)| = \sum_{i=1}^{N} \lambda_i |x_{i0}| = \|x_0\| \leq \alpha$ and $\alpha > k/c$.

## ESTIMATES OF DOMAINS OF ATTRACTION

Neural networks of the type considered herein have many equilibrium points. If a given equilibrium is asymptotically stable, or exponentially stable, then the extent of this stability is of interest. As usual, we assume that $x = 0$ is the equilibrium of interest. If $\phi(t, t_0, x_0)$ denotes a solution of the network (1) with $\phi(t_0, t_0, x_0) = x_0$, then we would like to know for which points $x_0$ it is true that $\phi(t, t_0, x_0)$ tends to the origin as $t \to \infty$. The set of all such points $x_0$ makes up the *domain of attraction* (the *basin of attraction*) of the equilibrium $x = 0$. In general, one cannot determine such a domain in its entirety. However, several techniques have been devised to estimate subsets of a domain of attraction. We apply one such method to neural networks, making use of Theorem 1. This technique is applicable to our other results as well, by making appropriate modifications.

We assume that the hypotheses (A-1), (A-2) and (A-3) are satisfied and for the free subsystem (2) we choose the Lyapunov function

$$v_i(p_i) = \frac{1}{2} p_i^2. \tag{9}$$

Then $Dv_{i_{(2)}}(p_i) \leq (-b_i + a_{ii})p_i^2$, $|p_i| < r_i$ for some $r_i > 0$. If (A-3) is satisfied, we must have $(-b_i + a_{ii}) < 0$ and $Dv_{i_{(2)}}(p_i)$ is negative definite over $B(r_i)$.

Let $C_{v_{0i}} = \{p_i \epsilon R : v_i(p_i) = \frac{1}{2} p_i^2 < \frac{1}{2} r_i^2 \triangleq v_{0i}\}$. Then $C_{v_{0i}}$ is contained in the domain of attraction of the equilibrium $p_i = 0$ for the free subsystem (2).

To obtain an estimate for the domain of attraction of $x = 0$ for the whole neural network (1), we use the Lyapunov function

$$v(x) = \sum_{i=1}^{N} \frac{1}{2}\alpha_i x_i^2 = \sum_{i=1}^{N} \alpha_i v_i(x_i). \qquad (10)$$

It is now an easy matter to show that the set

$$C_\lambda = \{x\epsilon R^N : v(x) = \sum_{i=1}^{N} \alpha_i v_i(x_i) < \lambda\}$$

will be a subset of the domain of attraction of $x = 0$ for the neural network (1), where

$$\lambda = \min_{1\le i\le N}(\alpha_i v_{0i}) = \min_{1\le i\le N}\left(\frac{1}{2}\alpha_i r_i^2\right).$$

In order to obtain the best estimate of the domain of attraction of $x = 0$ by the present method, we must choose the $\alpha_i$ in an optimal fashion. The reader is referred to the literature[9,13,14] where several methods to accomplish this are discussed.

## INSTABILITY RESULTS

Some of the equilibrium points in a neural network may be unstable. We present here a sample instability theorem which may be viewed as a counterpart to Theorem 2. Instability results, formulated as counterparts to other stability results of the type considered herein may be obtained by making appropriate modifications.

**(A-8)** For system (1), the interconnections satisfy the estimates

$$x_i A_{ii} G_i(x_i) \le \delta_i A_{ii} x_i^2,$$
$$|x_i A_{ij} G_j(x_j)| \le |x_i||A_{ij}|\sigma_{j2}|x_i|, \quad i \ne j$$

where $\delta_i = \sigma_{i1}$ when $A_{ii} < 0$ and $\delta_i = \sigma_{i2}$ when $A_{ii} > 0$ for all $|x_i| < r_i$, and for all $|x_j| < r_j, i,j = 1,\ldots,N$.

**(A-9)** The successive principal minors of the $N \times N$ test matrix $D = [d_{ij}]$ given by

$$d_{ij} = \begin{cases} \sigma_i, & i = j \\ -|A_{ij}|, & i \ne j \end{cases}$$

are positive, where $\sigma_i = \frac{b_i}{\sigma_{i2}} - A_{ii}$ when $i\epsilon F_s$ (i.e., stable subsystems) and $\sigma_i = -\frac{b_i}{\sigma_{i1}} + A_{ii}$ when $i\epsilon F_u$ (i.e., unstable subsystems) with $F = F_s \cup F_u$ and $F = \{1,\ldots,N\}$ and $F_u \ne \phi$.

We are now in a position to prove the following result.

**Theorem 4** *The equilibrium $x = 0$ of the neural network (1) is **unstable** if hypotheses (A-1), (A-8) and (A-9) are satisfied. If in addition, $F_s = \phi$ ($\phi$ denotes the empty set), then the equilibrium $x = 0$ is **completely unstable**.*

**Proof.** We choose the Lyapunov function

$$v(x) = \sum_{i \epsilon F_u} \alpha_i(-|x_i|) + \sum_{i \epsilon F_s} \alpha_i|x_i| \tag{11}$$

where $\alpha_i > 0$, $i = 1, \ldots, N$. Along the solutions of (1) we have (following the proof of Theorem 2), $Dv_{(1)}(x) \leq -\alpha^T Dw$ for all $x \epsilon B(r)$, $r = \min_i r_i$ where $\alpha^T = (\alpha_1, \ldots, \alpha_N)$, $D$ is defined in (A-9), and $w^T = \left[\frac{G_1(x_1)}{x_1}|x_1|, \ldots, \frac{G_N(x_N)}{x_N}|x_N|\right]$. We conclude that $Dv_{(1)}(x)$ is negative definite over $B(r)$. Since every neighborhood of the origin $x = 0$ contains at least one point $x'$ where $v(x') < 0$, it follows that the equilibrium $x = 0$ for (1) is unstable. Moreover, when $F_s = \phi$, then the function $v(x)$ is negative definite and the equilibrium $x = 0$ of (1) is in fact completely unstable (c.f. Chapter 5 in Ref. 7).

## STABILITY UNDER STRUCTURAL PERTURBATIONS

In specific applications involving adaptive schemes for learning algorithms in neural networks, the interconnection patterns (and external inputs) are changed to yield an evolution of different sets of desired asymptotically stable equilibrium points with appropriate domains of attraction. The present diagonal dominance conditions (see, e.g., hypothesis (A-6)) can be used as constraints to guarantee that the desired equilibria always have the desired stability properties.

To be more specific, we assume that a given neural network has been designed with a set of interconnections whose strengths can be varied from zero to some specified values. We express this by writing in place of (1),

$$\dot{x}_i = -b_i x_i + \sum_{j=1}^{N} \theta_{ij} \, A_{ij} \, G_j(x_j) + U_i(t), \quad for \ i = 1, \ldots, N, \tag{12}$$

where $0 \leq \theta_{ij} \leq 1$. We also assume that in the given neural network things have been arranged in such a manner that for some given desired value $\Delta > 0$, it is true that $\Delta = \min_i \left(\frac{b_i}{\sigma_{i2}} - \theta_{ii} A_{ii}\right)$. From what has been said previously, it should now be clear that if $U_i(t) \equiv 0$, $i = 1, \ldots, N$ and if the diagonal dominance conditions

$$\Delta - \sum_{\substack{j=1 \\ i \neq j}}^{N} \left(\frac{\lambda_j}{\lambda_i}\right) |\theta_{ij} A_{ij}| > 0, \quad for \ i = 1, \ldots, N \tag{13}$$

are satisfied for some $\lambda_i > 0$, $i = 1, \ldots, N$, then the equilibrium $x = 0$ for (12) will be asymptotically stable. It is important to recognize that condition (13) constitutes a single stability condition for the neural network under structural perturbations. Thus, the strengths of interconnections of the neural network may be rearranged in any manner to achieve some desired set of equilibrium points. If (13) is satisfied, then these equilibria will be asymptotically stable. (Stability under structural perturbations is nicely surveyed in Ref. 15.)

## CONCLUDING REMARKS

In the present paper we surveyed and applied results from the qualitative theory of large scale interconnected dynamical systems in order to develop a qualitative theory for neural networks of the Hopfield type. Our results are local and use as much information as possible in the analysis of a given equilibrium. In doing so, we established criteria for the exponential stability, asymptotic stability, and instability of an equilibrium in such networks. We also devised methods for estimating the domain of attraction of an asymptotically stable equilibrium and for estimating trajectory bounds for such networks. Furthermore, we showed that our stability results are applicable to systems under structural perturbations (e.g., as experienced in neural networks in adaptive learning schemes).

In arriving at the above results, we viewed neural networks as an interconnection of many single neurons, and we phrased our results in terms of the qualitative properties of the free single neurons and in terms of the network interconnecting structure. This viewpoint is particularly well suited for the study of hierarchical structures which naturally lend themselves to implementations[16] in VLSI. Furthermore, this type of approach makes it possible to circumvent difficulties which usually arise in the analysis and synthesis of complex high dimensional systems.

## Footnotes

[1]The work of A. N. Michel and J. A. Farrell was supported by NSF under grant ECS84-19918.

[2]The work of W. Porod was supported by ONR under grant N00014-86-K-0506.

## REFERENCES

[1] For a review, see, *Neural Networks for Computing*, J. S. Denker, Editor, American Institute of Physics Conference Proceedings **151**, Snowbird, Utah, 1986.

[2] J. J. Hopfield and D. W. Tank, *Science* **233**, 625 (1986).

[3] J. J. Hopfield, *Proc. Natl. Acad. Sci. U.S.A.* **79**, 2554 (1982), and *ibid.* **81**, 3088 (1984).

[4] G. E. Hinton and J. A. Anderson, Editors, *Parallel Models of Associative Memory*, Erlbaum, 1981.

[5] T. Kohonen, *Self-Organization and Associative Memory*, Springer-Verlag, 1984.

[6] A. N. Michel and R. K. Miller, *Qualitative Analysis of Large Scale Dynamical Systems*, Academic Press, 1977.

[7] R. K. Miller and A. N. Michel, *Ordinary Differential Equations*, Academic Press, 1982.

[8] I. W. Sandberg, *Bell System Tech. J.* **48**, 35 (1969).

[9] A. N. Michel, *IEEE Trans. on Automatic Control* **28**, 639 (1983).

[10] A. N. Michel, J. A. Farrell, and W. Porod, submitted for publication.

[11] J.-H. Li, A. N. Michel, and W. Porod, *IEEE Trans. Circ. and Syst.*, in press.

[12] G. A. Carpenter, M. A. Cohen, and S. Grossberg, *Science* **235**, 1226 (1987).

[13] M. A. Pai, *Power System Stability*, Amsterdam, North Holland, 1981.

[14] A. N. Michel, N. R. Sarabudla, and R. K. Miller, *Circuits, Systems and Signal Processing* **1**, 171 (1982).

[15] Lj. T. Grujic, A. A. Martynyuk and M. Ribbens-Pavella, *Stability of Large-Scale Systems Under Structural and Singular Perturbations*, Nauka Dumka, Kiev, 1984.

[16] D. K. Ferry and W. Porod, *Superlattices and Microstructures* **2**, 41 (1986).
